# Two is better than one: distinct roles for familiarity and recollection in retrieving palimpsest memories

**Cristina Savin**[1]
cs664@cam.ac.uk

**Peter Dayan**[2]
dayan@gatsby.ucl.ac.uk

**Máté Lengyel**[1]
m.lengyel@eng.cam.ac.uk

[1]Computational & Biological Learning Lab, Dept. of Engineering, University of Cambridge, UK
[2]Gatsby Computational Neuroscience Unit, University College London, UK

## Abstract

Storing a new pattern in a palimpsest memory system comes at the cost of interfering with the memory traces of previously stored items. Knowing the age of a pattern thus becomes critical for recalling it faithfully. This implies that there should be a tight coupling between estimates of age, as a form of *familiarity*, and the neural dynamics of recollection, something which current theories omit. Using a normative model of autoassociative memory, we show that a dual memory system, consisting of two interacting modules for familiarity and recollection, has best performance for both recollection and recognition. This finding provides a new window onto actively contentious psychological and neural aspects of recognition memory.

## 1   Introduction

Episodic memory such as that in the hippocampus acts like a palimpsest – each new entity to be stored is overlaid on top of its predecessors, and, in turn, is submerged by its successors. This implies both anterograde interference (existing memories hinder the processing of new ones) and retrograde interference (new memories overwrite information about old ones). Both pose important challenges for the storage and retrieval of information in neural circuits. Some aspects of these challenges have been addressed in two theoretical frameworks – one focusing on anterograde interference through the interaction of novelty and storage [1]; the other on retrograde interference in individual synapses [2]. However, neither fully considered the critical issue of retrieval from palimpsests; this is our focus.

First, [1] made the critical observation that autoassociative memories only work if normal recall dynamics are suppressed on presentation of new patterns that need to be stored. Otherwise, rather than memorizing the new pattern, the memory associated with the existing pattern that most closely matches the new input will be strengthened. This suggests that it is critical to have a mechanism for assessing pattern novelty or, conversely, familiarity, a function that is often ascribed to neocortical areas surrounding the hippocampus.

Second, [2] considered the palimpsest problem of overwriting information in synapses whose efficacies have limited dynamic ranges. They pointed out that this can be at least partially addressed through allowing multiple internal states (for instance forming a cascade) for each observable synaptic efficacy level. However, although [2] provide an attractive formalism for analyzing and optimizing synaptic storage, a retrieval mechanism associated with this storage is missing.

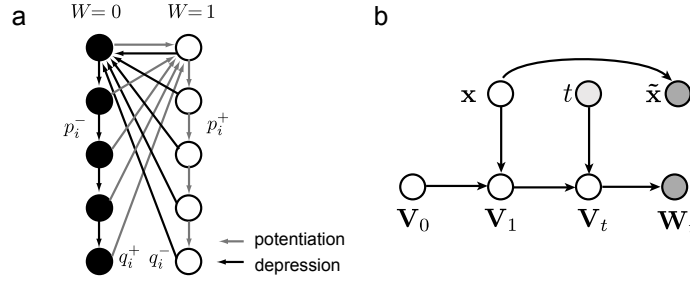

Figure 1: **a.** The cascade model. Internal states of a synapse (circles) can express one of two different efficacies (W, columns). Transitions between states are stochastic and can either be potentiating, or depressing, depending on pre- and postsynaptic activities. Probabilities of transitions between states expressing the same efficacy $p$ and between states expressing different efficacies, $q$, decrease geometrically with cascade depth. **b.** Generative model for the autoassociative memory task. The recall cue $\tilde{\mathbf{x}}$ is a noisy version of one of the stored patterns $\mathbf{x}$. Upon storing pattern $\mathbf{x}$ synaptic states changed from $\mathbf{V}_0$ (sampled from the stationary distribution of synaptic dynamics) to $\mathbf{V}_1$. Recall occurs after the presentation of $t - 1$ intervening patterns, when synapses are in states $\mathbf{V}_t$, with corresponding synaptic efficacies $\mathbf{W}_t$. Only $\mathbf{W}_t$ and $\tilde{\mathbf{x}}$ are observed at recall.

Although these pieces of work might seem completely unrelated, we show here that they are closely linked via retrieval. The critical fact about recall from memory, in general, is to know how the information should appear at the time of retrieval. In the case of a palimpsest, the trace of a memory in the synaptic efficacies depends critically on the *age* of the memory, i.e., its relative familiarity. This suggests a central role for novelty (or familiarity) signals during recollection. Indeed, we show retrieval is substantially worse when familiarity is not explicitly represented than when it is.

Dual system models for recognition memory are the topic of a heated debate [3, 4]. Our results could provide a computational rationale for them, showing that separating a perirhinal-like network (involved in familiarity) from a hippocampal-like network can be beneficial even when the only task is recollection. We also show that the task of recognition can also be best accomplished by combining the outputs of both networks, as suggested experimentally [4].

## 2   Storage in a palimpsest memory

We consider the task of autoassociative recall of binary patterns from a palimpsest memory. Specifically, the neural circuit consists of $N$ binary neurons that enjoy all-to-all connectivity. During storage, network activity is clamped to the presented pattern $\mathbf{x}$, inducing changes in the synapses' 'internal' states $\mathbf{V}$ and corresponding observed binary efficacies $\mathbf{W}$ (Fig. 1a).

At recall, we seek to retrieve a pattern $\mathbf{x}$ that was originally stored, given a noisy cue $\tilde{\mathbf{x}}$ and the current weight matrix $\mathbf{W}$. This weight matrix is assumed to result from storing $\mathbf{x}$ on top of the stationary distribution of the synaptic efficacies coming from the large number of patterns that had been previously stored, and then subsequently storing a sequence of $t - 1$ other intervening patterns with the same statistics on top of $\mathbf{x}$ (Fig. 1b).

In more detail, a pattern to be stored has density $f$, and is drawn from the distribution:

$$P_{\text{store}}(\mathbf{x}) = \prod_i P_{\text{store}}(x_i) = \prod_i f^{x_i} \cdot (1 - f)^{1 - x_i} \tag{1}$$

The recall cue is a noisy version of the original pattern, modeled using a binary symmetric channel:

$$P_{\text{noise}}(\tilde{\mathbf{x}}|\mathbf{x}) = \prod_i P_{\text{noise}}(\tilde{x}_i|x_i) \tag{2}$$

$$P_{\text{noise}}(\tilde{x}_i|x_i) = \left((1 - r)^{x_i} \cdot r^{1 - x_i}\right)^{\tilde{x}_i} \cdot \left(r^{x_i} \cdot (1 - r)^{1 - x_i}\right)^{1 - \tilde{x}_i} \tag{3}$$

where $r$ defines the level of input noise.

The recall time $t$ is assumed to come from a geometric distribution with mean $\bar{t}$:

$$\mathrm{P}_{\mathrm{recall}}(t) = \frac{1}{\bar{t}} \cdot \left(1 - \frac{1}{\bar{t}}\right)^{t-1} \tag{4}$$

The synaptic learning rule is local and stochastic, with the probability of an event actually leading to state changes determined by the current state of the synapse $V_{ij}$ and the activity at the pre- and post-synaptic neurons, $x_i$ and $x_j$. Hence, learning is specified through a set of transition matrices $\mathbf{M}(x_i, x_j)$, with $M(x_i, x_j)_{l'l} = \mathrm{P}(V'_{ij} = l'|V_{ij} = l, x_i, x_j)$. For convenience, we adopted the cascade model [2] (Fig. 1a), which assumes that the probability of potentiation and depression decays with cascade depth $i$ as a geometric progression, $q_i^{\pm} = \chi^{i-1}$, with $q_n^{\pm} = \frac{\chi^{n-1}}{1-\chi}$ to compensate for boundary effects. The transition between metastates is given by $p_i^{\pm} = \varsigma_{\pm} \frac{\chi^i}{1-\chi}$, with the correction factors $\varsigma_+ = \frac{1-f}{f}$ and $\varsigma_- = \frac{f}{1-f}$ ensuring that different metastates are equally occupied for different pattern sparseness values $f$ [2]. Furthermore, we assume synaptic changes occur only when the postsynaptic neuron is active, leading to potentiation if the presynaptic neuron is also active and to depression otherwise. The specific form of the learning rule could influence the memory span of the network, but we expect it not to change the results below qualitatively.

The evolution of the distribution over synaptic states after encoding can be described by a Markov process, with a transition matrix $\overline{\mathbf{M}}$ given as the average change in synaptic states expected after storing an arbitrary pattern from the prior $\mathrm{P}_{\mathrm{store}}(\mathbf{x})$, $\overline{\mathbf{M}} = \sum_{x_i, x_j} \mathrm{P}_{\mathrm{store}}(x_i) \cdot \mathrm{P}_{\mathrm{store}}(x_j) \cdot \mathbf{M}(x_i, x_j)$.

Additionally, we define the column vectors $\boldsymbol{\pi}^{\mathrm{V}}(x_i, x_j)$ and $\boldsymbol{\pi}^{\mathrm{W}}(x_i, x_j)$ for the distribution of the synaptic states and observable efficacies, respectively, when one of the patterns stored was $(x_i, x_j)$, such that $\pi_l^{\mathrm{W}}(x_i, x_j) = \mathrm{P}(W_{ij} = l|x_i, x_j)$ and $\pi_l^{\mathrm{V}}(x_i, x_j) = \mathrm{P}(V_{ij} = l|x_i, x_j)$. Given these definitions, we can express the final distribution over synaptic states as:

$$\boldsymbol{\pi}^{\mathrm{V}}(x_i, x_j) = \sum_t \left( \mathrm{P}_{\mathrm{recall}}(t) \cdot \overline{\mathbf{M}}^{t-1} \cdot \mathbf{M}(x_i, x_j) \cdot \boldsymbol{\pi}^{\infty} \right) \tag{5}$$

where we start from the stationary distribution $\boldsymbol{\pi}^{\infty}$ (the eigenvector of $\overline{\mathbf{M}}$ for eigenvalue 1), encode pattern $(x_i, x_j)$ and then $t - 1$ additional patterns from the same distribution. The corresponding weight distribution is $\boldsymbol{\pi}^{\mathrm{W}}(x_i, x_j) = \mathbf{T} \cdot \boldsymbol{\pi}^{\mathrm{V}}(x_i, x_j)$, where $\mathbf{T}$ is a $2 \times 2n$ matrix defining the deterministic mapping from synaptic states to observable efficacies.

The fact that the recency of the pattern to be recalled, $t$, appears in equation 5 implies that pattern age will strongly influence information retrieval. In the following, we consider two possible solutions to this problem. We first show the limitations of recall dynamics that involve a single, monolithic module which averages over $t$. We then prove the benefits of a dual system with two qualitatively different modules, one of which explicitly represents an estimate of pattern age.

## 3 A single module recollection system

### 3.1 Optimal retrieval dynamics

Since information storage by synaptic plasticity is lossy, the recollection task described above is a probabilistic inference problem [5,6]. Essentially, neural dynamics should represent (aspects of) the posterior over stored patterns, $\mathrm{P}(\mathbf{x}|\tilde{\mathbf{x}}, \mathbf{W})$, that expresses the probability of any pattern $\mathbf{x}$ being the correct response for the recall query given a noisy recall cue, $\tilde{\mathbf{x}}$, and the synaptic efficacies $\mathbf{W}$.

In more detail, the posterior over possible stored patterns can be computed as:

$$\mathrm{P}(\mathbf{x}|\mathbf{W}, \tilde{\mathbf{x}}) \propto \mathrm{P}_{\mathrm{store}}(\mathbf{x}) \cdot \mathrm{P}_{\mathrm{noise}}(\tilde{\mathbf{x}}|\mathbf{x}) \cdot \mathrm{P}(\mathbf{W}|\mathbf{x}) \tag{6}$$

where we assume that evidence from the weights factorizes over synapses[1], $\mathrm{P}(\mathbf{W}|\mathbf{x}) = \prod_{ij} \mathrm{P}(W_{ij}|x_i, x_j)$.

Previous Bayesian recall dynamics derivations assumed learning rules for which the contribution of each pattern to the final weight were the same, irrespective of the order of pattern presentation [5,6]. By contrast, the Markov chain behaviour of our synaptic learning rule forces us to explicitly consider pattern age. Furthermore, as pattern age is unknown at recall, we need to integrate over all possible $t$ values (Eq. 5). This integral (which is technically a sum, for discrete $t$) can be computed analytically using the eigenvalue decomposition of the transition matrix $\overline{\mathbf{M}}$. Alternatively, if the value of $t$ is known during recall, the prior is replaced by a delta function, $\mathrm{P}_{\mathrm{recall}}(t) = \delta(t - t^*)$.

There are several possible ways of representing the posterior in Eq.6 through neural dynamics without reifying $t$. For consistency, we assume neural states to be binary, with network activity at each step representing a sample from the posterior [7, 8]. An advantage of this approach is that the full posterior is represented in the network dynamics, such that higher decision modules can not only extract the 'best' pattern (for the mean squared error cost function considered here, this would be the mean of the posterior) but also estimate the uncertainty of this solution. Nevertheless, other representations, for example representing the parameters of a mean-field approximation to the true posterior [5, 9, 10], would also be possible and similarly informative about uncertainty.

In particular, we use Gibbs sampling, as it allows for neurally plausible recall dynamics [7]. This results in asynchronous updates, in which the activity of a neuron $x_i$ changes stochastically as a function of its input cue $\tilde{x}_i$, the activity of all other neurons, $\mathbf{x}_{\setminus i}$, and neighbouring synapses, $\mathbf{W}_{i,\cdot}$ and $\mathbf{W}_{\cdot,i}$. Specifically, the Gibbs sampler results in a sigmoid transfer function, with the total current to the neuron given by the log-odds ratio:

$$I_i^{\mathrm{rec}} \quad = \quad \log \frac{\mathrm{P}(x_i = 1 | \mathbf{x}_{\setminus i}, \mathbf{W}, \tilde{x}_i)}{\mathrm{P}(x_i = 0 | \mathbf{x}_{\setminus i}, \mathbf{W}, \tilde{x}_i)} = I_i^{\mathrm{rec,in}} + I_i^{\mathrm{rec,out}} + a\tilde{x}_i + b \qquad (7)$$

with the terms $I_{\mathrm{rec}}^{\mathrm{in/out}}$ defining the evidence from the incoming and outgoing synapses of neuron $i$, and the constants $a$ and $b$ determined by the prior over patterns and the noise model.[2] The terms describing the contribution from recurrent interactions, have a similar shape:

$$I_i^{\mathrm{rec,in}} \quad = \quad \sum_j \left( c_1^{\mathrm{in}} \cdot W_{ij}\, x_j + c_2^{\mathrm{in}} \cdot W_{ij} + c_3^{\mathrm{in}} \cdot x_j + c_4^{\mathrm{in}} \right) \qquad (8)$$

$$I_i^{\mathrm{rec,out}} \quad = \quad \sum_j \left( c_1^{\mathrm{out}} \cdot W_{ji}\, x_j + c_2^{\mathrm{out}} \cdot W_{ji} + c_3^{\mathrm{out}} \cdot x_j + c_4^{\mathrm{out}} \right) \qquad (9)$$

The parameters $c_k^{\mathrm{in/out}}$, uniquely determined by the learning rule and the priors for $\mathbf{x}$ and $t$, rescale the contribution of the evidence from the weights as a function of pattern age (see supplementary text). Furthermore, these constants translate into a unique signal, giving a sort of 'sufficient statistic' for the expected memory strength. Note that the optimal dynamics include two homeostatic processes, corresponding to global inhibition, $\sum_j x_j$, and neuronal excitability regulation, $\sum_j W_{ij}$, that stabilize network activity during recall.

## 3.2 Limitations

Beside the effects of assuming a factorized weight distribution, the neural dynamics derived above should be the best we can do given the available data (i.e. recall cue and synaptic weights). How well does the network fare in practice?

Performance is as expected when pattern age is assumed known: as the available information from the weights decreases, so does performance, finally converging to control levels, defined by the retrieval performance of a network without plastic recurrent connections, i.e. when inference uses only the recall cue and the prior over stored patterns (Fig. 2a, green). When $t$ is unknown, performance also deteriorates with increasing pattern age, however this time beneath control levels (Fig. 2a, blue). Intuitively, one can see that relying on the prior over $t$ is similar to assuming $t$ fixed to a value close

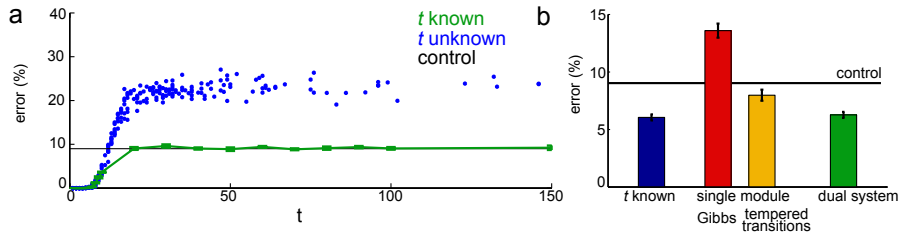

Figure 2: **a.** Recall performance for a single module memory system. **b.** Average recollection error comparison for the single and dual memory system. Black lines mark control performance, when ignoring the information from the synaptic weights.

to the mean of this prior. When the pattern that was actually presented is older than this estimate, the resulting memory signal is weaker than expected, suggesting that the initial pattern was very sparse (since a pair of inactive elements does not induce any synaptic changes according to our learning rule). However, less reasonable is the fact that averaging over the prior distribution of recall times $t$ (Eq. 4), performance is worse than this control (Fig. 2b).

One possible reason for this failure is that the sampling procedure used for inference might not work in certain cases. Since Gibbs samplers are known to mix poorly when the shape of the posterior is complex (with strong correlations, as in frustrated Ising models), perhaps our neural dynamics are unable to sample the desired distribution effectively. We confirmed this hypothesis by implementing a more sophisticated sampling procedure using tempered transitions [11] (details in supplementary text). Indeed, with tempered transitions performance becomes significantly better than control, even for the cases where Gibbs sampling fails (Fig. 2b). Unfortunately, there has yet to be a convincing suggestion as to how tempering dynamics (or in fact any other sampling algorithm that works well with correlated posteriors) can be represented neurally since, for example, they require a global acceptance decision to be taken at the end of each temperature cycle.

It is worth noting that with more complex synaptic dynamics (e.g. deeper cascades) simple Gibbs sampling works reasonably well (data not shown), probably because the posterior is smoother and hence easier to sample.

## 4   A dual memory system

An alternative to implicitly marginalizing over the age of the pattern throughout the inference process is to estimate it at the same time as performing recollection. This suggests the use of dual modules that together estimate the joint posterior $P(\mathbf{x}, t|\tilde{\mathbf{x}}, \mathbf{W})$, with sampling proceeding in a loop: the familiarity module generates a sample from the posterior over the age of the currently estimated pattern, $P(t|\mathbf{x}, \tilde{\mathbf{x}}, \mathbf{W})$; and the recollection module uses this estimated age to compute a new sample from the distribution over possible stored patterns given the age, $P(\mathbf{x}|\tilde{\mathbf{x}}, \mathbf{W}, t)$ (Fig. 3a).

The module that computes familiarity can also be seen as a palimpsest, with each pattern overlaying, and being overlaid by, its predecessors and successors. Formally, it needs to compute the probability $P(t|\mathbf{x}, \tilde{\mathbf{x}}, \mathbf{W})$, as the system continues to implement a Gibbs sampler with $t$ as an additional dimension. As a separate module, the neural network estimating familiarity cannot however access the weights $\mathbf{W}$ of the recollection module. A biologically plausible approximation is to assume that the familiarity module uses a separate set of weights, which we call $\mathbf{W}^{\mathrm{fam}}$. Also, it is clear from Fig. 1b that $t$ is independent of $\tilde{\mathbf{x}}$ conditioned on $\mathbf{x}$, thus the conditioning on $\tilde{\mathbf{x}}$ can be dropped when computing the posterior over $t$, that is, external input need only feed directly into the recollection but not the familiarity module (Fig. 3a).

In particular, we assume a feedforward network structure in the familiarity module, with each neuron receiving the output of the recollection module as inputs through synapses $\mathbf{W}^{\mathrm{fam}}$. These synaptic

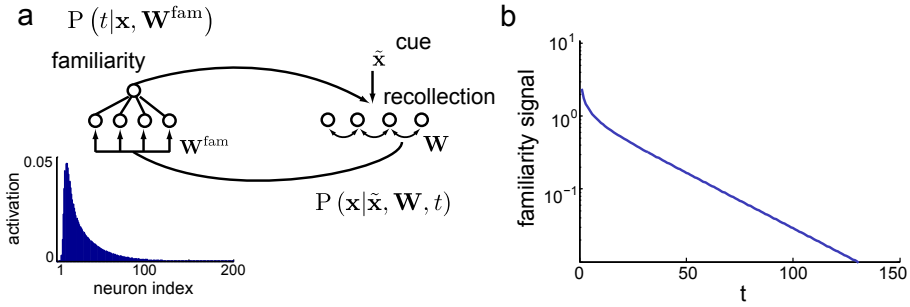

Figure 3: **a.** An overview of the dual memory system. The familiarity network has a feedforward structure, with the activity of individual neurons estimating the probability of the true pattern age being a certain value $t$, see example in inset. The estimated pattern age translates into a familiarity signal, which scales the contribution of the recurrent inputs in the network dynamics. **b.** Dependence of the familiarity signal on the estimated pattern age.

weights change according to the same cascade rule used for recollection.[3] For simplicity, we assume that the familiarity neurons are always activated during encoding, so that synapses can change state (either by potentiation or depression) with every storage event.

Concretely, the familiarity module consists of $N_{\text{fam}}$ neurons, each corresponding to a certain pattern age in the range $1$–$N_{\text{fam}}$ (the last unit codes for $t \geq N_{\text{fam}}$). This forms a localist code for familiarity. The total input to a neuron is given by the log-posterior $I_i^{\text{fam}} = \log \text{P}(t = i | \mathbf{x}, \mathbf{W}^{\text{fam}})$ which translates into a simple linear activation function:

$$I_i^{\text{fam}} = \sum_j \left[ c_{1,i}^{\text{fam}} W_{ij}^{\text{fam}} x_j + c_{2,i}^{\text{fam}} W_{ij}^{\text{fam}} + c_{3,i}^{\text{fam}} x_j + c_{4,i}^{\text{fam}} \right] + \log \text{P}(t) - \log(Z) \qquad (10)$$

where the constants $c_{k,i}^{\text{fam}}$ are similar to parameters $c^{\text{in/out}}$ before (albeit different for each neuron because of their tuning to different values of $t$), and $Z$ is the unknown partition function.

As mentioned above, we treat the activity of the familiarity module as a sample from the posterior over age $t$. This representation requires lateral competition between different units such that only one can become active at each step. Dynamics of this sort can be implemented using a softmax operator, $\text{P}(x_i^{\text{fam}} = 1) = \frac{e^{I_i}}{\sum_j e^{I_j}}$ (thus rendering the evaluation of the partition function $Z$ unnecessary), and are a common feature of a range of neural models [12, 13].

Critically, this familiarity module is not just a convenient theoretical construct associated with retrieval. First, as we mentioned before, the assessment of novelty actually plays a key part in memory *storage* – in making the decision as to whether a pattern that is presented is novel, and so should be stored, or familiar, and so should have its details be recalled. This venerable suggestion [1] has played a central part in the understanding of structure-function relationships in the hippocampus. The graded familiarity module that we have suggested is an obvious extension of this idea; the use for retrieval is new. Second, it is in general accord with substantial data on the role of perirhinal cortex and the activity of neurons in this structure [3]. Recency neurons would be associated with small values of $t$; novelty neurons with large or effectively infinite values of $t$ [14], although perirhinal cortex appears to adopt a population coding strategy for age, rather than just one-of-$n$.

The recollection module has the same dynamics as before, with constants $c_i$ computed assuming $t$ fixed to the output of the familiarity module. Thus we predict that familiarity multiplicatively modulates recurrent interactions in the recollection module during recall. Since there is a deterministic mapping between $t$ and this modulatory factor (Fig. 3b), it can be computed using a linear unit pooling the outputs of all the neurons in the familiarity module, with weights given by the corresponding values for $c_i^{\text{fam}}(t)$.

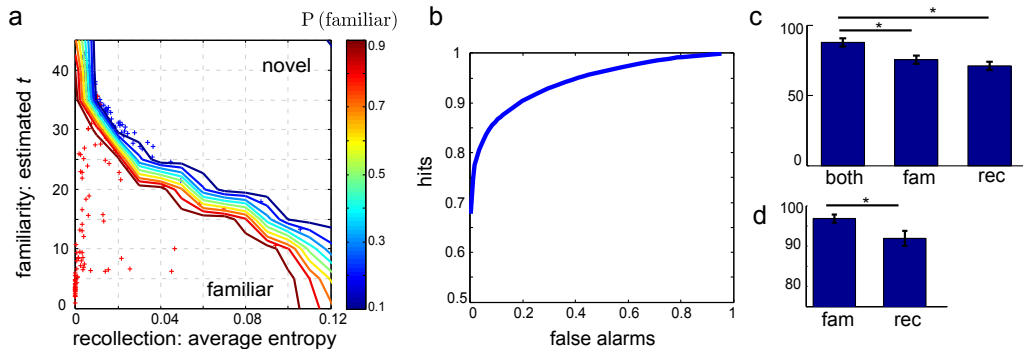

Figure 4: **a.** Decision boundaries for the recognition module. **b.** Corresponding ROC curve. **c.** Performance comparison when the decision layer uses signals from the familiarity module, the recollection module, or both. **d.** Same comparison, when data is restricted to recent stimuli. Note that difference between fam and rec became significant compared to **c**.

In order to compare single and dual module systems fairly, the computational resources employed by each should be the same. We therefore reduced the overall connectivity in the dual system such that the two have the same total number of synapses. Moreover, since elements of $\mathbf{W}^{\mathrm{fam}}$ are correlated, the effective number of connections is in fact somewhat lower in the dual system. Regardless, the dual memory system performs significantly better than the single module system (Fig. 2b).

## 5 Recognition memory

We have so far considered familiarity merely as an instrument for effective recollection. However, there are many practical and experimental tasks in which it is sufficient to make a binary decision about whether a pattern is novel or familiar rather than recalling it in all its gory detail. It is these tasks that have been used to elucidate the role of perirhinal cortex in recognition memory.

In the dual module system, information about recognition is available from both the familiarity module (patterns judged to have young ages are recognized) and the recollection module (patterns recalled with higher certainty are recognized). We therefore construct an additional decision module which takes the outputs of the familiarity and recollection modules and maps them into a binary behavioral response (familiar vs. novel).

Specifically, we use the average of the entropies associated with the activities of neurons in the recollection module and the mean estimate of $t$ from the familiarity module. Since the palimpsest property implicitly assumes that all patterns have been presented at some point, we define a pattern to be familiar if its age is less than a fixed threshold $t_{\mathrm{th}}$. We train the decision module using a Gaussian process classifier[4] [15], which yields as outcome the probability of a hit, $\mathrm{P}(\mathrm{familiar}|t^*, \mathbf{x}^*)$, shown in Fig. 4a. The shape of the resulting discriminator, that it is not parallel to either axis, suggests that the output of both modules is needed for successful recognition, as suggested experimentally [4, 16]. The fact that a classifier trained using only one of the two dimensions cannot match the recognition performance of that using both confirms this observation (Fig. 4c).

Moreover, the ROC curve produced by the classifier, plotting hit rates against false alarms as relative losses are varied, has a similar shape to those obtained for human behavioral data: it has a so-called 'curvi-linear' character because of the apparent intersect at a finite hit probability for 0 false alarm rate [17] (Fig. 4b). Lastly, as recognition is known to rely more on familiarity for relatively recent patterns [18], we estimate recognition performance for recent patterns, which we define as having age $t \leq \frac{t_{\mathrm{th}}}{2}$. To determine the contribution of each module in recognition outcomes in this case, we estimate performance of classifiers trained on single input dimensions for this test data. Consistent with experimental data, our analysis reveals that the familiarity signal gives a more reliable estimate of novelty, compared to the recollection output for relatively recent items (Fig. 4d).

# 6 Conclusions and discussion

Knowing the age of a pattern is critical for retrieval from palimpsest memories, a consideration that has so far eluded theoretical inquiry. We showed that a memory system could either treat this information implicitly, by marginalizing over all possible ages, or it could estimate age explicitly as a form of familiarity. In principle, both solutions should have similar performance, given the same resources. In practice, however, a system involving dual modules is significantly better.

In our model, the posterior over possible stored patterns was represented in neural activities via samples. We showed that a complex, biologically-questionable sampling procedure would be necessary for the implicit, single module, system. Instead, a dual memory system with two functionally distinct but closely interacting modules, yielded the best performance both for efficient recollection and for recognition. Importantly, though Gibbs sampling and tempered transitions provide a useful framework for understanding the performance differences between different memory systems, the presented results are not restricted to a sampling-based implementation. Since age and identity are tightly correlated, a mean field solution that use factorized distributions [5] shows very similar behavior (see supplementary text). Similarly, the specific details of the familiarity module are not critical for these effects, which should be apparent for any alternative implementation correctly estimating pattern age.

Representing pattern age, $t$, explicitly essentially amounts to implementing an auxiliary variable for sampling the space of possible patterns, $\mathbf{x}$ more efficiently. Such auxiliary variable methods are widely used to increase sampling efficiency when other, simpler methods fail [19]. Moreover, since $t$ in our case specifically modulates the correlated components of the posterior it can be seen as a 'temperature' parameter, and so we can understand the advantages brought about by the dual system as due to implementing a form of 'simulated tempering' – a class of methods known to help mixing in strongly correlated posteriors.

Our proposal provides a powerful new window onto the contentious debate about the neural mechanisms of recognition and recall. The rationale for our familiarity network was improving recollection; however, the form of the network was motivated by the substantial experimental data [14] on recognition, and indeed standard models of perirhinal cortex activity [20]. These, for instance, also rely on some form of inhibition to mediate interactions between different familiarity neurons. Nevertheless, our model is the first to link the computational function of familiarity networks to recall; it is distinct also in that it considers palimpsest synapses, as previous models use purely additive learning rules [20]. Although we only considered pattern age as the basis of familiarity here, the principle of the interaction between familiarity and recollection remains the same in an extended setting, when familiarity characterizes the expected strength of the memory trace more completely, including the effects of retention interval, number of repetitions, and spacing between repetitions. Future work with the extended model should allow us to address familiarity, novelty, and recency neurons in the perirhinal cortex, and indeed provide a foundation for new thinking about this region.

In our model familiarity interacts with recollection by multiplicatively (or divisively) modulating the contribution of recurrent inputs in the recollection module. Neurally, this effect could be mediated by shunting inhibition via specific classes of hippocampal interneurons which target the dendritic segment corresponding to recurrent connections, thus rescaling the relative contribution of external versus recurrent inputs [21]. Whether pathways reaching CA3 from perirhinal cortex through entorhinal cortex preserve a sufficient amount of input specificity of feed-forward inhibition is unknown.

Our theory predicts important systems-level aspects of memory from synaptic-level constraints. In particular, by optimizing our dual system solely for memory recall we also predicted non-trivial ROC curves for recognition that are in at least broad qualitative agreement with experiments. Future work will be needed to explore whether the ROC curves in our model show similar dissociations in response to specific lesions of the two modules to those found in recent experiments [22, 23] and the relation to other recognition memory models [24].

## Acknowledgements

This work was supported by the Wellcome Trust (CS, ML) and the Gatsby Charitable Foundation (PD).

## Footnotes

[1]This assumption is never exactly true in practice, as synapses that share a pre- or post- synaptic partner are bound to be correlated. Here, we assume the intervening patterns cause independent weight changes and ignore the effects of such correlations.

[2]Real neurons can only receive information from their presynaptic partners, so cannot estimate $I_{\mathrm{rec}}^{\mathrm{out}}$. We therefore ran simulations without this term in the dynamics and found that although it did decrease recall performance, this decrease was similar to that obtained by randomly pruning half of the connections in the network and keeping this term in the dynamics (not shown). This indicated that performance is mostly determined by the number of available synapses used for inference, and not so much by the direction of those synapses. Hence, in the following we use both terms and leave the systematic study of connectivity for future work.

[3]There is nothing to say that the learning rule that optimizes the recollection network's ability to recall patterns should be equally appropriate for assessing familiarity. Hence, the familiarity module could have their own learning rule, optimized for its specific task.

[4]The specific classifier was chosen as it allows for an easy estimation of the ROC curves. Future work should explore analytical decision rules.

# References

[1] Hasselmo, M.E. The role of acetylcholine in learning and memory. *Current opinion in neurobiology* **16**, 710–715 (2006).

[2] Fusi, S., Drew, P.J. & Abbott, L.F. Cascade models of synaptically stored memories. *Neuron* **45**, 599–611 (2005).

[3] Brown, M.W. & Aggleton, J.P. Recognition memory: What are the roles of the perirhinal cortex and hippocampus? *Nature Reviews Neuroscience* **2**, 51–61 (2001).

[4] Wixted, J.T. & Squire, L.R. The medial temporal lobe and the attributes of memory. *Trends in Cognitive Sciences* **15**, 210–217 (2011).

[5] Sommer, F.T. & Dayan, P. Bayesian retrieval in associative memories with storage errors. *IEEE transactions on neural networks* **9**, 705–713 (1998).

[6] Lengyel, M., Kwag, J., Paulsen, O. & Dayan, P. Matching storage and recall: hippocampal spike timing-dependent plasticity and phase response curves. *Nature Neuroscience* **8**, 1677–1683 (2005).

[7] Ackley, D., Hinton, G. & Sejnowski, T. A learning algorithm for Boltzmann machines. *Cognitive Science* **9**, 147–169 (1995).

[8] Fiser, J., Berkes, P., Orbán, G. & Lengyel, M. Statistically optimal perception and learning: from behavior to neural representations. *Trends in Cognitive Sciences* **14**, 119–130 (2010).

[9] Hinton, G. Deterministic Boltzmann learning performs steepest descent in weight-space. *Neural Computation* **1**, 143–150 (1990).

[10] Lengyel, M. & Dayan, P. Uncertainty, phase and oscillatory hippocampal recall. *Advances in Neural Information Processing* (2007).

[11] Neal, R.M. Sampling from multimodal distributions using tempered transitions. *Statistics and Computing* **6**, 353–366 (1996).

[12] Fukai, T. & Tanaka, S. A simple neural network exhibiting selective activation of neuronal ensembles: from winner-take-all to winners-share-all. *Neural computation* **9**, 77–97 (1997).

[13] Bogacz, R. & Gurney, K. The basal ganglia and cortex implement optimal decision making between alternative actions. *Neural computation* **19**, 442–477 (2007).

[14] Xiang, J.Z. & Brown, M.W. Differential neuronal encoding of novelty, familiarity and recency in regions of the anterior temporal lobe. *Neuropharmacology* **37**, 657–676 (1998).

[15] Rasmussen, C.E. & Williams, C.K.I. *Gaussian Processes for Machine Learning* (MIT Press, 2006).

[16] Warburton, E.C. & Brown, M.W. Findings from animals concerning when interactions between perirhinal cortex, hippocampus and medial prefrontal cortex are necessary for recognition memory. *Neuropsychologia* **48**, 2262–2272 (2010).

[17] Yonelinas, A.P. Components of episodic memory: the contribution of recollection and familiarity. *Philosophical Transactions of the Royal Society B: Biological Sciences* **356**, 1363–1374 (2001).

[18] Yonelinas, A. The nature of recollection and familiarity: A review of 30 years of research. *Journal of memory and language* **46**, 441–517 (2002).

[19] Iba, Y. Extended ensemble Monte Carlo. *Int. J. Mod. Phys* **12**, 653–656 (2001).

[20] Bogacz, R. Comparison of computational models of familiarity discrimination in the perirhinal cortex. *Hippocampus* (2003).

[21] Mitchell, S. Shunting inhibition modulates neuronal gain during synaptic excitation. *Neuron* (2003).

[22] Fortin, N.J., Wright, S.P. & Eichenbaum, H. Recollection-like memory retrieval in rats is dependent on the hippocampus. *Nature* **431**, 188–191 (2004).

[23] Cowell, R., Winters, B., Bussey, T. & Saksida, L. Paradoxical false memory for objects after brain damage. *Science* (2010).

[24] Norman, K. & O'Reilly, R. Modeling hippocampal and neocortical contributions to recognition memory: A complementary-learning-systems approach. *Psychological Review* (2003).

